# Learning a Tree of Metrics
# with Disjoint Visual Features

**Sung Ju Hwang**
University of Texas
Austin, TX 78701
sjhwang@cs.utexas.edu

**Kristen Grauman**
University of Texas
Austin, TX 78701
grauman@cs.utexas.edu

**Fei Sha**
University of Southern California
Los Angeles, CA 90089
feisha@usc.edu

## Abstract

We introduce an approach to learn discriminative visual representations while exploiting external semantic knowledge about object category relationships. Given a hierarchical taxonomy that captures semantic similarity between the objects, we learn a corresponding *tree of metrics* (ToM). In this tree, we have one metric for each non-leaf node of the object hierarchy, and each metric is responsible for discriminating among its immediate subcategory children. Specifically, a Mahalanobis metric learned for a given node must satisfy the appropriate (dis)similarity constraints generated only among its subtree members' training instances. To further exploit the semantics, we introduce a novel regularizer coupling the metrics that prefers a sparse *disjoint* set of features to be selected for each metric relative to its ancestor (supercategory) nodes' metrics. Intuitively, this reflects that visual cues most useful to distinguish the generic classes (e.g., feline vs. canine) should be *different* than those cues most useful to distinguish their component fine-grained classes (e.g., Persian cat vs. Siamese cat). We validate our approach with multiple image datasets using the WordNet taxonomy, show its advantages over alternative metric learning approaches, and analyze the meaning of attribute features selected by our algorithm.

## 1   Introduction

Visual recognition is a fundamental computer vision problem that demands sophisticated image representations—due to both the large number of object categories a system should ultimately recognize, as well as the noisy cluttered conditions in which training examples are often captured. The research community has made great strides in recent years by training discriminative models with an array of well-engineered descriptors, e.g., capturing gradient texture, color, or local part configurations. In particular, recent work shows promising results when integrating powerful feature selection techniques, whether through kernel combination [1, 2], sparse coding dictionaries [3], structured sparsity regularization [4, 5], or metric learning approaches [6, 7, 8, 9, 10].

However, typically the semantic information embedded in the learned features is restricted to the category labels on image exemplars. For example, a learned metric generates (dis)similarity constraints using instances with the different/same class label; multiple kernel learning methods optimize feature weights to minimize class prediction errors; group sparsity regularizers exploit class labels to guide the selected dimensions. Unfortunately, this means richer information about the *meaning* of the target object categories is withheld from the learned representations. While sufficient for objects starkly different in appearance, this omission is likely restrictive for objects with finer-grained distinctions, or when a large number of classes densely populate the original feature space.

We propose a metric learning approach to learn discriminative visual representations while also exploiting external knowledge about the target objects' semantic similarity.[1] We assume the external knowledge itself is available in the form of a hierarchical taxonomy over the objects (e.g., from WordNet or some other knowledge base). Our approach exploits these semantics in two novel ways.

First, we construct a *tree of metrics* (ToM) to directly capture the hierarchical structure. In this tree, each metric is responsible for discriminating among its immediate object subcategories. Specifically, we learn one metric for each non-leaf node, and require it to satisfy (dis)similarity constraints generated among its subtree members' training instances. We use a variant of the large-margin nearest neighbor objective [11], and augment it with a regularizer for sparsity in order to unify Mahalanobis parameter learning with a simple means of feature selection.

Second, rather than learn the metrics at each node independently, we introduce a novel regularizer for *disjoint sparsity* that couples each metric with those of its ancestors. This regularizer specifies that a disjoint set of features should be selected for a given node and its ancestors, respectively. Intuitively, this represents that the visual features most useful to distinguish the coarse-grained classes (e.g., feline vs. canine) should often be *different* than those cues most useful to distinguish their fine-grained subclasses (e.g., Persian vs. Siamese cat, German Shepherd vs. Boxer). The resulting optimization problem is convex, and can be optimized with a projected subgradient approach.

The ideas of exploiting label hierarchy and model sparsity are not completely new to computer vision and machine learning researchers. Hierarchical classifiers are used to speed up classification time and alleviate data sparsity problems [12, 13, 14, 15, 16]. Parameter sparsity is increasingly used to derive parsimonious models with informative features [4, 5, 3].

Our novel contribution lies in the idea of ToM and disjoint sparsity together as a new strategy for visual feature learning. Our idea reaps the benefits of both schools of thought. Rather than relying on learners to discover both sparse features and a visual hierarchy fully automatically, we use external "real-world" knowledge expressed in hierarchical structures to *bias* which sparsity patterns we want the learned discriminative feature representations to exhibit. Thus, our end-goal is not any sparsity pattern returned by learners, but the patterns that are in concert with rich high-level semantics.

We validate our approach with the Animals with Attributes [17] and ImageNet [18] datasets using the WordNet taxonomy. We demonstrate that the proposed ToM outperforms both global and multiple-metric metric learning baselines that have similar objectives but lack the hierarchical structure and proposed disjoint sparsity regularizer. In addition, we show that when the dimensions of the original feature space are interpretable (nameable) visual attributes, the disjoint features selected for super- and sub-classes by our method can be quite intuitive.

## 2 Related Work

A wide variety of feature learning approaches have been explored for visual recognition. Some of the very best results on benchmark image classification tasks today use multiple kernel learning approaches [1, 2] or sparse coding dictionaries for local features (e.g., [3]). One way to regularize visual feature selection is to prefer that object categories *share* features, so as to speed up object detection [19]; more recent work uses group sparsity to impose some sharing among the (un)selected features within an object category or view [4, 5]. We instead seek disjoint features between coarse and fine categories, such that the regularizer helps to focus on useful differences across levels.

Metric learning has been a subject of extensive research in recent years, in both vision and learning. Good visual metrics can be trained with boosting [20, 21], feature weight learning [6], or Mahalanobis metric learning methods [7, 8, 10]. An array of Mahalanobis metric learners has been developed in the machine learning literature [22, 23, 11]. The idea of using multiple "local" metrics to cover a complex feature space is not new [24, 9, 10, 25]; however, in contrast to our approach, existing methods resort to clustering or (flat) class labels to determine the partitioning of training instances to metrics. Most methods treat the partitioning and metric learning processes separately, but some recent work integrates the grouping directly into the learning objective [21], or trains mul-

tiple metrics jointly across tasks [26]. No previous work explores mapping the semantic hierarchy to a ToM, nor couples metrics across the hierarchy levels, as we propose. To show the impact, in experiments we directly compare to a state-of-the-art approach for learning multiple metrics.

Previous metric learning work integrates feature learning and selection via a regularizer for sparsity [27], as we do here. However, whereas that approach targets sparsity in the linear transformed space, ours targets sparsity in the original feature space, and, most importantly, also includes a disjoint sparsity regularizer. The advantage in doing so is that our learner will be able to return both discriminative and interpretable feature dimensions, as we demonstrate in our results. Transformed feature spaces—while suitably flexible if only discriminative power is desired—add layers that complicate interpretability, not only to models for individual classifiers but also (more seriously) to tease apart patterns across related categories (such as parent-child).

The "orthogonal transfer" by [28] is most closely related in spirit to our goal of selecting disjoint features. However, unlike [28], our regularizer will yield truly disjoint features when minimized—a property hinging on the metric-based classification scheme we have chosen. Our learning problem is guaranteed to be convex, whereas hyperparameters need to be tuned to ensure convexity in [28]. We return to these differences in Section 3.3, after explaining our algorithm in detail.

External semantics beyond object class labels are rarely used in today's object recognition systems, but recent work has begun to investigate new ways to integrate richer knowledge. Hierarchical taxonomies have natural appeal, and researchers have studied ways to discover such structure automatically [29, 30, 13], or to integrate known structure to train classifiers at different levels [12, 31]. The emphasis is generally on saving prediction time (by traversing the tree from its root) or combining decisions, whereas we propose to influence feature learning based on these semantics. While semantic structure need not always translate into helping visual feature selection, the correlation between WordNet semantics and visual confusions observed in [32] supports our use of the knowledge base in this work. The machine learning community has also long explored hierarchical classification (e.g., [14, 15, 16]). Of this work, our goals most relate to [14], but our focus is on learning features discriminatively and biasing toward a disjoint feature set via regularization.

Beyond taxonomies, researchers are also injecting semantics by learning mid-level nameable "attributes" for object categorization (e.g., [17, 33]). We show that when our method is applied to attributes as base features, the disjoint sparsity effects appear to be fairly interpretable.

## 3   Approach

We review briefly the techniques for learning distance metrics. We then describe an $\ell_1$-norm based regularization for selecting a sparse set of features in the context of metric learning. Building on that, we proceed to describe our main algorithmic contribution, that is, the design of a metric learning algorithm that prefers not only *sparse* but also *disjoint* features for discriminating different categories.

### 3.1   Distance metric learning

Many learning algorithms depend on calculating distances between samples, notably $k$-nearest neighbor classifiers or clustering. While the default is to use the Euclidean distance, the more general Mahalanobis metric is often more suitable. For two data points $\boldsymbol{x}_i, \boldsymbol{x}_j \in \mathbb{R}^{\mathrm{D}}$, their (squared) Mahalanobis distance is given by

$$d_{\boldsymbol{M}}^2(\boldsymbol{x}_i, \boldsymbol{x}_j) = (\boldsymbol{x}_i - \boldsymbol{x}_j)^{\mathrm{T}}\boldsymbol{M}(\boldsymbol{x}_i - \boldsymbol{x}_j), \tag{1}$$

where $\boldsymbol{M}$ is a positive semidefinite matrix $\boldsymbol{M} \succeq 0$. Arguably, the Mahalanobis distance can better model complex data, as it considers correlations between feature dimensions.

Learning the optimal $\boldsymbol{M}$ from labeled data has been an active research topic (e.g., [23, 22, 11]). Most methods follow an intuitively appealing strategy: a good metric $\boldsymbol{M}$ should pull data points belonging to the same class closer and push away data points belonging to different classes. As an illustrative example, we describe the technique used in constructing large margin nearest neighbor (LMNN) classifiers [11], to which our empirical studies extensively compare.

In LMNN, each point $\boldsymbol{x}_i$ in the training set is associated with two sets of different data points in $\boldsymbol{x}_i$'s nearest neighbors (identified in the Euclidean distance): the "targets" whose labels are the same as

$x_i$'s and the "impostors" whose labels are different. Let $x_i^+$ denote the "target" and $x_i^-$ denote the "impostor" sets, respectively. LMNN identifies the optimal $M$ as the solution to,

$$\min_{M \succeq 0} \quad \ell(M) = \sum_i \sum_{j \in x_i^+} d_M^2(x_i, x_j) + \gamma \sum_{ijl} \xi_{ijl}$$

$$\text{subject to} \quad 1 + d_M^2(x_i, x_j) - d_M^2(x_i, x_l) \le \xi_{ijl}; \; \xi_{ijl} \ge 0 \; . \forall \, j \in x_i^+, \; l \in x_i^- \qquad (2)$$

where the objective function $\ell(M)$ balances two forces: pulling the target towards $x_i$ and pushing the impostor away. The latter is characterized by the constraint composed of a triplet of data points: the distance to an impostor should be greater than the distance to a target by at least a margin of 1, possibly with the help of a slack variable $\xi_{ijl}$. The minimization of eq. (2) is a convex optimization problem with semidefinite constraints on $M \succeq 0$, and is tractable with standard techniques.

Our approach extends previous work on metric learning in two aspects: i) we apply a sparsity-based regularization to identify informative features (Section 3.2); ii) at the same time, we seek metrics that rely on *disjoint* subsets of features for categories at different semantic granularities (Section 3.3).

## 3.2 Sparse feature selection for metric learning

*How can we learn a metric such that only a sparse set of features are relevant?* Examining the definition of the Mahalanobis distance in eq. (1), we deduce that if the $d$-th feature of $x$ is not to be used, it is sufficient and necessary for the $d$-th diagonal element of $M$ be zero.

Therefore, analogous to the use of $\ell_1$-norm by the popular LASSO technique [34], we add the $\ell_1$-norm of $M$'s diagonal elements to the large margin metric learning criterion $\ell(M)$ in eq. (2),

$$\min_{M \succeq 0} \quad \sum_i \sum_{j \in x_i^+} d_M^2(x_i, x_j) + \gamma \sum_{ijl} \xi_{ijl} + \lambda \text{Trace}[M], \qquad (3)$$

where we have omitted the constraints as they are not changed. $\lambda$ and $\gamma$ are nonnegative (hyper)parameters trading off the sparsity of the model and the other parts in the objective. Note that since the matrix trace $\text{Trace}[\cdot]$ is a linear function of its argument, this sparse feature metric learning problem remains a convex optimization.

## 3.3 Learning a tree of metrics (ToM) with disjoint visual features

*How can we learn a tree of metrics so each metric uses features disjoint from its ancestors'?*

**Using disjoint features** To characterize the "disjointness" between two metrics $M_t$ and $M_{t'}$, we use the vectors of their nonnegative diagonal elements $v_t$ and $v_{t'}$ as proxies to which features are (more heavily) used. This is a reasonable choice as we use the sparsity-inducing $\ell_1$-norm in learning the metrics. We measure their degree of "competition" for common features,

$$C(M_t, M_{t'}) = \|v_t + v_{t'}\|_2^2 \, . \qquad (4)$$

Intuitively, if a feature dimension is not used by either metric, the competition for that feature is low. If a feature dimension is used by both metrics heavily, then the competition is high. Consequently, minimizing eq. (4) as a regularization term will encourage different metrics to use disjoint features. Note that the measure is a convex function of $v_t$ and $v_{t'}$, hence also convex in $M_t$ and $M_{t'}$.

**Learning a tree of metrics** Formally, assume we have a tree $\mathcal{T}$ where each node corresponds to a category. Let $t$ index the $\mathsf{T}$ *non-leaf* or internal nodes. We learn a metric $M_t$ to differentiate its children categories $c(t)$. For any node $t$, we use $\mathcal{D}(t)$ to denote those training samples whose labeled categories are offspring of $t$, and $a(t)$ to denote the nodes on the path from the root to $t$.

To learn our metrics $\{\boldsymbol{M}_t\}_{t=1}^{\mathsf{T}}$, we apply similar strategies of learning metrics for large-margin nearest neighbor classifiers. We cast it as a convex optimization problem:

$$\min_{\{\boldsymbol{M}_t\}\succeq 0} \quad \sum_t \sum_{c\in c(t)} \sum_{i,j\in\mathcal{D}(c)} d_{\boldsymbol{M}_t}^2(\boldsymbol{x}_i,\boldsymbol{x}_j) + \gamma \sum_{t,c,r,ijl} \xi_{tcrijl} + \sum_t \lambda_t \mathsf{Trace}[\boldsymbol{M}_t]$$
$$+ \sum_t \sum_{a\in a(t)} \gamma_{ta} C(\boldsymbol{M}_t, \boldsymbol{M}_a) \qquad (5)$$
$$\text{subject to} \quad \forall\, t, \forall\, c\in c(t), \forall\, r\in c(t)-\{c\}, \forall\, \boldsymbol{x}_i, \boldsymbol{x}_j \in \mathcal{D}(c),\ \boldsymbol{x}_l \in \mathcal{D}(r)$$
$$1 + d_{\boldsymbol{M}_t}^2(\boldsymbol{x}_i,\boldsymbol{x}_j) - d_{\boldsymbol{M}_t}^2(\boldsymbol{x}_i,\boldsymbol{x}_l) \le \xi_{tcrijl};\ \xi_{tcrijl} \ge 0\,.$$

In short, there are $\mathsf{T}$ learning (sub)problems, one for each metric. Each metric learning problem is in the style of the sparse feature metric learning eq. (3). However, more importantly, these metric learning problems are *coupled* together through the disjoint regularization. Our disjoint regularization encourages a metric $\boldsymbol{M}_t$ to use different sets of features from its *super*-categories—categories on the tree path from the root.

**Numerical optimization** The optimization problem in eq. (5) is convex, though nonsmooth due to the nonnegative slack variables. We use the subgradient method, previously used for similar problems [11]. For problems with a large taxonomy, learning all the regularization coefficients $\lambda_t$ and $\gamma_{ta}$ is prohibitive, as the number of coefficient combinations is $O(k^{\mathsf{T}})$, where $\mathsf{T}$ is the number of nodes and $k$ is the number of values a coefficient can take. Thus, for the large-scale problems we focus on, we use a simpler and computationally more efficient strategy of Sequential Optimization (SO) by sequentially optimizing one metric at a time. Specifically, we optimize the metric at the root node and then its children, assuming the metric at the root is fixed. We then recursively (in breadth-first-search) optimize the rest of the metrics, always treating the metrics at the higher level of the hierarchy as fixed. This strategy has a significantly reduced computational cost of $O(k\mathsf{T})$. In addition, the SO procedure allows each metric to be optimized with different parameters and prevents a badly-learned low-level metric from influencing upper-level ones through the disjoint regularization terms. (This can also be achieved by adjusting all regularization coefficients in parallel through extensive cross-validation, but at a much higher computational expense.)

**Using a tree of metrics for classification** Once the metrics at all nodes are learned, they can be used for several classification tasks (e.g., with $k$-NN or as a kernel to a SVM). In this work, we study two tasks in particular: 1) We consider "per-node classification", where the metric at each node is used to discriminate its sub-categories. Since decisions at higher-level nodes must span a variety of object sub-categories, these generic decisions are interesting to test the learned features in a broader context. 2) We consider hierarchical classification [35], a natural way to use the full ToM. In this case, we examine the recognition accuracy for the finest-level categories only. We classify an object from the root node down; the leaf node that terminates the path is the predicted label.

We stress that our metric learning criterion of eq. (5) aims to minimize classification errors at each node. Thus, improvement in per-node accuracy is more directly indicative of whether the learning has resulted in useful metrics. Understanding the relation between per-node and full multi-class accuracy has been a challenging research problem in building hierarchical classifiers [16, 12].

**Relationship to orthogonal transfer** Our work shares a similar spirit to the "orthogonal transfer" idea explored in [28]. The authors there use non-overlapping features to construct multiple SVM classifiers for hierarchical classification of text documents. Concretely, they propose an orthogonal regularizer $\sum_{ij} K_{ij}|\boldsymbol{w}_i^\mathsf{T}\boldsymbol{w}_j|$ where $\boldsymbol{w}_i$ and $\boldsymbol{w}_j$ are the SVM parameters. Minimizing it will reduce the similarity of the parameter vectors and make them "orthogonal" to each other. However, orthogonality does not necessarily imply disjoint features. This can be seen with a contrived two-dimensional counterexample where $\boldsymbol{w}_i = [1\ \ -1]^\mathsf{T}$ and $\boldsymbol{w}_j = [-1\ \ -1]^\mathsf{T}$. Both features are used, yet the two parameter vectors are orthogonal. In contrast, our disjoint regularizer eq. (4) is more indicative of true disjointness. Specifically, when our regularizer attains its minimum value of zero, we are guaranteed that features are non-overlapping as our $\boldsymbol{v}_i$ and $\boldsymbol{v}_j$ are *nonnegative* diagonal elements of positive semidefinite matrices. Our regularizer is also guaranteed to be convex, whereas the convexity of the regularizer in [28] depends critically on tuning $K_{ij}$.

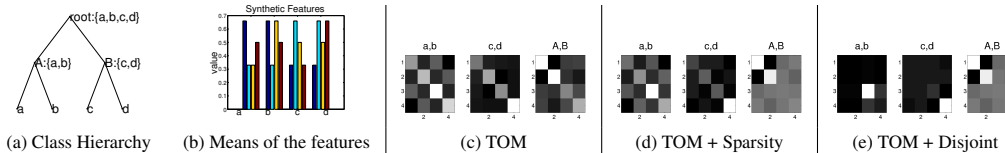

|  (a) Class Hierarchy | (b) Means of the features | (c) TOM | (d) TOM + Sparsity | (e) TOM + Disjoint |

Figure 1: Synthetic dataset example. Our disjoint regularizer yields a sparse metric that only considers the feature dimension(s) necessary for discrimination at that given level.

## 4 Results

We validate our ToM approach on several datasets, and consider three baselines: 1) **Euclidean**: Euclidean distance in the original feature space, 2) **Global LMNN**: a single global metric for all classes learned with the LMNN algorithm [11], and 3) **Multi-Metric LMNN**: one metric learned per class using the multiple metric LMNN variant [11]. We use the code provided by the authors.

To evaluate the influence of each aspect of our method, we test it under three variants: 1) **ToM**: ToM learning without any regularization terms, 2) **ToM+Sparsity**: ToM learning with the sparsity regularization term, and 3) **ToM+Disjoint**: ToM learning with the disjoint regularization term. For all experiments, we test with five random data splits of 60%/20%/20% for train/validation/test. We use the validation data to set the regularization parameters $\lambda$ and $\gamma$ among candidate values $\{0, 1, 10, 100, 1000\}$, and we generate 500 $(\boldsymbol{x}_i, \boldsymbol{x}_j, \boldsymbol{x}_l)$ training triplets per class.

### 4.1 Proof of concept on synthetic dataset

First we use synthetic data to clearly illustrate disjoint sparsity regularization. We generate data with precisely the property that coarser categories are distinguishable using feature dimensions distinct from those needed to discriminate their subclasses. Specifically, we sample 2000 points from each of four 4D Gaussians, giving four leaf classes $\{a, b, c, d\}$. They are grouped into two superclasses $A = \{a, b\}$ and $B = \{c, d\}$. The first two dimensions of all points are specific to the superclass decision ($A$ vs. $B$), while the last two are specific to the subclasses. See Fig. 1 (a) and (b).

We run hierarchical $k$-nearest neighbor classification ($k = 3$) on the test set. ToM+Sparsity increases the recognition rate by $0.90\%$, while ToM+Disjoint increases it by $4.05\%$. Thus, as expected, disjoint sparsity does best, since it selects different features for the super- and sub-classes. Accordingly, in the learned Mahalanobis matrices for each node (Fig. 1(c)-(e)), we see disjoint sparsity zeros out the unneeded features for the upper-level metric, showed as black squares in the figure (e). In contrast, the ToM+Sparsity features are sub-optimal and fit to some noise in the data (d).

### 4.2 Visual recognition experiments

Next we demonstrate our approach on challenging visual recognition tasks.

**Datasets and implementation details** We validate with three datasets drawn from two publicly available image collections: Animals with Attributes (AWA) [17] and ImageNet [18, 32]. Both are well-suited for our scenario, since they consist of fine-grained categories that can be grouped into more general object categories. AWA contains 30,475 images and 50 animal classes, and we use it to create two datasets: 1) **AWA-PCA**, which uses the provided features (SIFT, rgSIFT, PHOG, SURF, LSS, RGB), concatenated, standardized, and PCA-reduced to 50 dimensions, and 2) **AWA-ATTR**, which uses 85-dimensional attribute predictions as the original feature space. The latter is formed by concatenating the outputs of 85 linear SVMs trained to predict the presence/absence of the 85 nameable properties annotated by [17], e.g., furry, white, quadrupedal, etc. For our third dataset **VEHICLE-20**, we take 20 vehicle classes and 26,624 images from ImageNet, and apply PCA to reduce the authors' provided visual word features [32] to 50 dimensions per image (The dimensionality worked best for the Global LMNN baseline.).

We use WordNet to generate the semantic hierarchies for all datasets. We retrieve all nodes in WordNet that contain any of the object class names on their word lists. In the case of homonyms, we manually disambiguate the word sense. Then, we build a compact partial hierarchy over those nodes by 1) pruning out any node that has only one child (i.e., removing superfluous nodes), and 2) resolving any instances of multiple parentship by choosing the path from the leaf to root having the most overlap with other classes. See Figures 2 and 3 for the resulting AWA and VEHICLE trees.

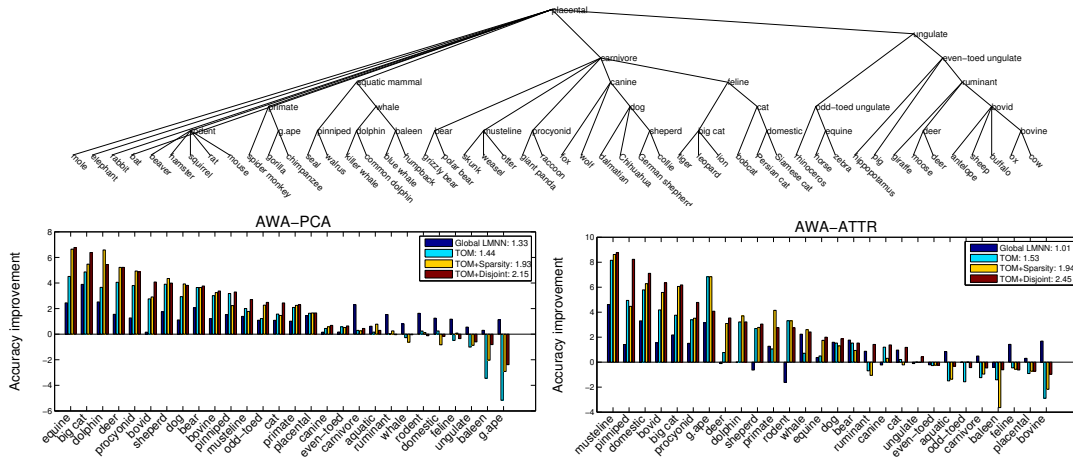

Figure 2: Semantic hierarchy for AWA (top) and the per-node accuracy improvements relative to Euclidean distance, for the AWA-PCA (left) and AWA-ATTR (right) datasets. Numbers in legends denote average improvement over all nodes. We generally achieve a sizable accuracy gain relative to the Global LMNN baseline (dark left bar for each class), showing the advantage of exploiting external semantics with our ToM approach.

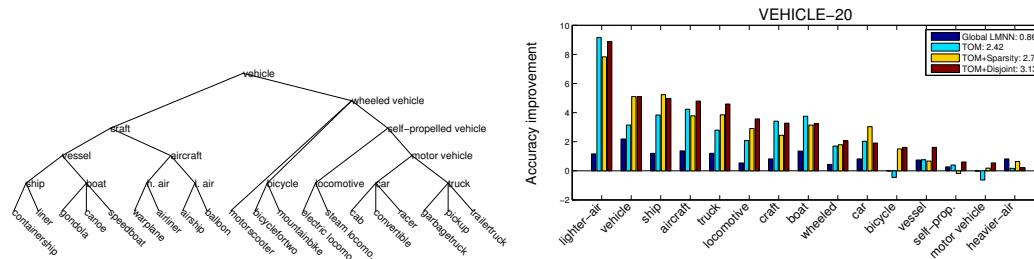

Figure 3: Semantic hierarchy for VEHICLE-20 and the per-node accuracy gains, plotted as above.

Throughout, we evaluate classification accuracy using $k$-nearest neighbors ($k$-NN). For ToM, at node $n$ we use $k = 2^{l_n - 1} + 1$, where $l_n$ is the level of the node, and $l_n = 1$ for leaf nodes. This means we use a larger $k$ at the higher nodes in the tree where there is larger intra-class variation, in an effort to be more robust to outliers. For the Euclidean and LMNN baselines, which lack a hierarchy, we simply use $k=3$. Note that ToM's setting at the final decision nodes (just above a leaf) is also $k = 3$, comparable to the baselines.

### 4.2.1 Per-node accuracy and analysis of the learned representations

Since our algorithm optimizes the metrics at every node, we first examine the resulting per-node decisions. That is, how accurately can we predict the correct subcategory at any given node? The bar charts in Figures 2 and 3 show the results, in terms of raw $k$-NN accuracy improvements over the Euclidean baseline. For reference, we also show the Global LMNN baseline. Multi-Metric LMNN is omitted here, since its metrics are only learned for the leaf node classes. We observe a good increase for most classes, as well as a clear advantage relative to LMNN. Furthermore, our results are usually strongest when including the novel disjoint sparsity regularizer. This result supports our basic claim about the potential advantage of exploiting external semantics in ToM.

We find that absolute gains are similar in either the PCA or ATTR feature spaces for AWA, though exact gains per class differ. While the ATTR variant exposes the semantic features directly to the learner, the PCA variant encapsulates an array of low-level descriptors into its dimensions. Thus, while we can better interpret the *meaning* of disjoint sparsity on the attributes, our positive result on raw image features assures that disjoint feature selection is also amenable in the more general case.

To look more closely at this, Table 1 displays representative superclasses from AWA-ATTR together with the attributes that ToM+Disjoint selects as discriminative for their subclasses. The attributes shown are those with nonzero weights in the learned metrics. Intuitively, we see that often the selected attributes are indeed useful for discriminating the child classes. For example, 'tusks' and 'plankton' attributes help distinguish common dolphins from killer whales, whereas 'stripes' and

| Superclass | Subclasses | Attributes selected | Superclass | Subclass | Attributes selected |
|---|---|---|---|---|---|
| dolphin | common dolphin, killer whale | tusks, plankton, blue, gray, red, patches, slow, muscle, active, insects | whale | dolphin, baleen whale | black, white, blue, gray, toughskin, chewteeth, strainteeth, smelly, slow, muscle, active, fish, hunter, skimmer, oldworld, arctic... |
| equine | horse, zebra | stripes, domestic, orange, red, yellow, toughskin, newworld, arctic, bush | odd-toed ungulate | equine, rhinoceros | fast, longneck, hairless, black, white, yellow, patches, spots, bulbous, longleg, buckteeth, horns, tusks, smelly... |

Table 1: Attributes selected by ToM+Disjoint for various superclass objects in AWA. See text.

| | AWA-ATTR | | AWA-PCA | | VEHICLE-20 | |
|---|---|---|---|---|---|---|
| Method | Correct label | Semantic similarity | Correct label | Semantic similarity | Correct label | Semantic similarity |
| Euclidean | $32.36 \pm 0.21$ | $53.60 \pm 0.26$ | $17.54 \pm 0.38$ | $38.11 \pm 0.58$ | $28.51 \pm 0.56$ | $56.10 \pm 0.41$ |
| Global LMNN | $32.49 \pm 0.42$ | $53.93 \pm 0.88$ | $\mathbf{19.62 \pm 0.51}$ | $40.34 \pm 0.32$ | $29.65 \pm 0.44$ | $57.57 \pm 0.45$ |
| Multi-metric LMNN | $32.34 \pm 0.35$ | $53.73 \pm 0.71$ | $17.61 \pm 0.33$ | $38.94 \pm 0.31$ | $30.00 \pm 0.51$ | $57.91 \pm 0.54$ |
| ToM | $36.79 \pm 0.27$ | $58.36 \pm 0.09$ | $18.70 \pm 0.41$ | $43.44 \pm 0.43$ | $31.23 \pm 0.67$ | $60.72 \pm 0.54$ |
| ToM + Sparsity | $37.58 \pm 0.32$ | $59.29 \pm 0.58$ | $18.79 \pm 0.46$ | $43.38 \pm 0.34$ | $32.09 \pm 0.18$ | $62.66 \pm 0.26$ |
| ToM + Disjoint | $\mathbf{38.29 \pm 0.61}$ | $\mathbf{59.72 \pm 0.62}$ | $19.00 \pm 0.30$ | $\mathbf{43.59 \pm 0.19}$ | $\mathbf{32.77 \pm 0.32}$ | $\mathbf{63.01 \pm 0.21}$ |

Table 2: Multi-class hierarchical classification accuracy and semantic similarity on all three datasets. Numbers are averages over 5 splits, and standard errors for 95% confidence interval. Our method outperforms the baselines in almost all cases, and notably provides more semantically close predictions. See text.

'domestic' help distinguish zebras from horses. At the same time, as desired, we see that the attributes useful for coarser-level categories are distinct from those employed to discriminate the finer-level objects. For example, 'fast', 'longneck', or 'hairless' are used to differentiate equine from rhino, but are excluded when differentiating horses from zebras (equine's subclasses).

### 4.2.2   Hierarchical multi-class classification accuracy

Next we evaluate the complete multi-class classification accuracy, where we use all the learned ToM metrics together to predict the leaf-node label of the test points. This is a 50-way task for AWA, and a 20-way task for VEHICLES. Table 2 shows the results.

We score accuracy in two ways: **Correct label** records the percentage of examples assigned the correct (leaf) label, while **Semantic similarity** records the semantic similarity between the predicted and true labels. For both, higher is better. The former is standard recognition accuracy, while the latter gives a more nuanced view of the "semantic magnitude" of the classifiers' errors. Specifically, we calculate the semantic similarity between classes (nodes) $i$ and $j$ using the metric defined in [36], which counts the number of nodes shared by their two parent branches, divided by the length of the longest of the two branches. In the spirit of other recent evaluations [37, 32, 36], this metric reflects that some errors are worse than others; for example, calling a Persian cat a Siamese cat is a less glaring error than calling a Persian cat a horse. This is especially relevant in our case, since our key motivation is to instill external semantics into the feature learning process.

In terms of pure label correctness, ToM improves over the strong LMNN baselines for both AWA-ATTR and VEHICLE-20. Further, in all cases, we see that disjoint sparsity is an important addition to ToM. However, in AWA-PCA, Global LMNN produces the best results by a statistically insignificant margin. We did not find a clear rationale for this one case. For AWA-ATTR, however, our method is substantially better than Global LMNN, perhaps due to our method's strength in exploiting semantic features. While we initially expected Multi-Metric LMNN to outperform Global LMNN, we suspect it struggles with clusters that are too close together. For all cases when ToM+Disjoint outperforms the LMNN or Euclidean baselines, the improvement is statistically significant.

In terms of semantic similarity, our ToM is better than all baselines on all datasets. This is a very encouraging result, since it suggests our approach is in fact leveraging semantics in a useful way. In practice, the ability to make such "reasonable" errors is likely to be increasingly important as the community tackles larger and larger multi-class recognition problems.

**Conclusion**   We presented a new metric learning approach for visual recognition that integrates external semantics about object hierarchy. Experiments with challenging datasets indicate its promise, and support our hypothesis that outside knowledge about how objects relate is valuable for feature learning. In future work, we are interested in exploring local features in this context, and considering ways to learn both the hierarchy and the useful features simultaneously.

**Acknowledgments**   F. Sha is supported by NSF IIS-1065243 and benefited from discussions with D. Zhou and B. Kulis. K. Grauman is supported by NSF IIS-1065390.

## Footnotes

[1]We use "learned representation" and "learned metric" interchangeably, since we deal with sparse Mahalanobis metrics, which are equivalent to selecting a subset of features and applying a linear feature space transformation.

# References

[1] A. Vedaldi, V. Gulshan, M. Varma, and A. Zisserman. Multiple kernels for object detection. In *ICCV*, 2009.

[2] P. Gehler and S. Nowozin. On feature combination for multiclass object classification. In *ICCV*, 2009.

[3] J. Yang, K. Yu, Y. Gong, and T. Huang. Linear spatial pyramid matching using sparse coding for image classification. In *CVPR*, 2009.

[4] L.-J. Li, H. Su, E. Xing, and L. Fei-Fei. Object bank: A high-level image representation for scene classification and semantic feature sparsification. In *NIPS*, 2010.

[5] Y. Jia, M. Salzmann, and T. Darrell. Factorized latent spaces with structured sparsity. In *NIPS*, 2010.

[6] A. Frome, Y. Singer, and J. Malik. Image retrieval and classification using local distance functions. In *NIPS*, 2006.

[7] P. Kumar, P. Torr, and A. Zisserman. An invariant large margin nearest neighbour classifier. In *ICCV*, 2007.

[8] P. Jain, B. Kulis, and K. Grauman. Fast image search for learned metrics. In *CVPR*, 2008.

[9] D. Ramanan and S. Baker. Local distance functions: A taxonomy, new algorithms, and an evaluation. In *PAMI*, 2011.

[10] Z. Wang, Y. Hu, and L.-T. Chia. Image-to-class distance metric learning for image classification. In *ECCV*, 2010.

[11] K. Q. Weinberger and K. L. Saul. Distance metric learning for large margin nearest neighbor classification. *JMLR*, 10:207–244, June 2009.

[12] M. Marszalek and C. Schmid. Constructing category hierarchies for visual recognition. In *ECCV*, 2008.

[13] G. Griffin and P. Perona. Learning and using taxonomies for fast visual category recognition. In *CVPR*, 2008.

[14] D. Koller and M. Sahami. Hierarchically classifying documents using very few words. In *ICML*, 1997.

[15] A. McCallum, R. Rosenfeld, T. Mitchell, and A. Ng. Improving text classification by shrinkage in a hierarchy of classes. In *ICML*, 1998.

[16] L. Cai and T. Hofmann. Hierarchical document categorization with support vector machines. In *CIKM*, 2004.

[17] C. Lampert, H. Nickisch, and S. Harmeling. Learning to detect unseen object classes by between-class attribute transfer. In *CVPR*, 2009.

[18] J. Deng, W. Dong, R. Socher, L.-J. Li, K. Li, and L. Fei-F ei. ImageNet: A large-scale hierarchical image database. In *CVPR*, 2009.

[19] A. Torralba and K. Murphy. Sharing visual features for multiclass and multiview object detection. *PAMI*, 29(5), 2007.

[20] G. Shakhnarovich. *Learning Task-Specific Similarity*. PhD thesis, MIT, 2006.

[21] B. Babenko, S. Branson, and S. Belongie. Similarity functions for categorization: from monolithic to category specific. In *ICCV*, 2009.

[22] A. Globerson and S. Roweis. Metric learning by collapsing classes. In *NIPS*, pages 451–458. 2006.

[23] J. Davis, B. Kulis, P. Jain, S. Sra, and I. Dhillon. Information-theoretic metric learning. In *ICML*, 2007.

[24] K. Weinberger and L. Saul. Fast solvers and efficient implementations for distance metric learning. In *ICML*, 2008.

[25] Q. Chen and S. Sun. Hierarchical large margin nearest neighbor classification. In *ICPR*, 2010.

[26] S. Parameswaran and K. Weinberger. Large margin multi-task metric learning. In *NIPS*, 2010.

[27] Y. Ying, K. Huang, and C. Campbell. Sparse metric learning via smooth optimization. In *NIPS*. 2009.

[28] D. Zhou, L. Xiao, and M. Wu. Hierarchical classification via orthogonal transfer. In *ICML*, 2011.

[29] J. Sivic, B. Russell, A. Zisserman, W. Freeman, and A. Efros. Unsupervised discovery of visual object class hierarchies. In *CVPR*, 2008.

[30] E. Bart, I. Porteous, P. Perona, and M. Welling. Unsupervised learning of visual taxonomies. In *CVPR*, 2008.

[31] A. Zweig and D. Weinshall. Exploiting object hierarchy: Combining models from different category levels. In *ICCV*, 2007.

[32] J. Deng, A. Berg, K. Li, and L. Fei-Fei. What does classifying more than 10,000 image categories tell us? In *ECCV*, 2010.

[33] Y. Wang and G. Mori. A discriminative latent model of object classes and attributes. In *ECCV*, 2010.

[34] R. Tibshirani. Regression shrinkage and selection via the lasso. *J. Roy. Statistical Society*, 58:267–288, 1994.

[35] S. Dumais and H. Chen. Hierarchical classification of web content. In *Research and Development in Information Retrieval*, 2000.

[36] R. Fergus, H. Bernal, Y. Weiss, and A. Torralba. Semantic label sharing for learning with many categories. In *ECCV*, 2010.

[37] K. Barnard, Q. Fan, R. Swaminathan, A. Hoogs, R. Collins, P. Rondot, and J. Kaufhold. Evaluation of localized semantics: data, methodology, and experiments. Technical report, University of Arizona, 2005.

